# Pedestrian-Centric 3D Pre-collision Pose and Shape Estimation from Dashcam Perspective

**Meijun Wang[1], Yu Meng[1][*], Zhongwei Qiu[2], Chao Zheng[1], Yan Xu[1], Xiaorui Peng[1], Jian Gao[3]**
[1]University of Science and Technology Beijing
[2]Alibaba DAMO Academy
[3]Northwest University
drwmj@xs.ustb.edu.cn, myu@ustb.edu.cn, qiuzhongwei.qzw@alibaba-inc.com
{miniflash, b20160225, pxrw}@xs.ustb.edu.cn, gaojian1@stumail.nwu.edu.cn

## Abstract

Pedestrian pre-collision pose is one of the key factors to determine the degree of pedestrian-vehicle injury in collision. Human pose estimation algorithm is an effective method to estimate pedestrian emergency pose from accident video. However, the pose estimation model trained by the existing daily human pose datasets has poor robustness under specific poses such as pedestrian pre-collision pose, and it is difficult to obtain human pose datasets in the wild scenes, especially lacking scarce data such as pedestrian pre-collision pose in traffic scenes. In this paper, we collect pedestrian-vehicle collision pose from the dashcam perspective of dashcam and construct the first Pedestrian-Vehicle Collision Pose dataset (PVCP) in a semi-automatic way, including 40k+ accident frames and 20K+ pedestrian pre-collision pose annotation (2D, 3D, Mesh). Further, we construct a Pedestrian Pre-collision Pose Estimation Network (PPSENet) to estimate the collision pose and shape sequence of pedestrians from pedestrian-vehicle accident videos. The PPSENet first estimates the 2D pose from the image (Image to Pose, ITP) and then lifts the 2D pose to 3D mesh (Pose to Mesh, PTM). Due to the small size of the dataset, we introduce a pre-training model that learns the human pose prior on a large number of pose datasets, and use iterative regression to estimate the pre-collision pose and shape of pedestrians. Further, we classify the pre-collision pose sequence and introduce pose class loss, which achieves the best accuracy compared with the existing relevant *state-of-the-art* methods. Code and data are available for research at https://github.com/wmj142326/PVCP.

## 1 Introduction

Pedestrian pre-collision pose refers to the emergency actions pedestrians take when facing potential hazards before collision accidents, affecting both the severity of pedestrian injuries and subsequent injury assessments (1). Using collision simulation software to reconstruct pedestrian-vehicle accidents is a popular and effective method for analyzing pedestrian injuries (2; 3; 4; 5). However, the current input for initial pose still relies on predefined gait sequence templates (6; 2; 3; 4; 5) or manual measurement of pose angles from accident images, the former cannot represent the posture of pedestrians in real accidents, and the latter is inefficient. Computer vision-based pedestrian pose estimation methods can directly estimate pose information such as joint positions or limb angles from images in real-time (7; 8; 9; 10; 11; 12; 13). Existing pose estimation methods are trained and applied on multiple datasets for various scenarios, adapting to different downstream tasks. However, unlike common poses, pedestrian pre-collision poses in traffic scenes are specific, with differences in spatial-

---

[*]Corresponding author.

temporal characteristics and challenges such as dynamic backgrounds, sudden scene changes, and occlusions of lower limbs (14; 15). Directly applying existing algorithms to pedestrian pre-collision pose estimation in traffic scenarios does not achieve perfect adaptation effects.

Training a proprietary network with specific datasets can effectively improve the pose estimation performance of the network in that scenario. Precise 2D pose annotations can be obtained through time and manpower-intensive efforts, indoor motion capture (Mocap) systems (16; 17) utilize markers and sensors to acquire high-quality 3D motion data. However, acquiring ground truth (GT) 3D joint positions in-the-wild is nearly impossible (12). Existing in-the-wild datasets either do not contain human pose (18; 19) labels or only include limited movements of daily activities (20; 21; 14). Furthermore, training a model using a large amount of data is costly, and video data of pedestrian-vehicle collisions belongs to small sample scarce data in traffic scenes, making dataset collection difficult. Dashcams or public surveillance devices are the only sources of data (22; 23; 24; 15), further constrains the approaches to dataset creation, thereby enhancing the difficulty and complexity of producing such datasets.

In this work, we constructed a Pedestrian-Vehicle Pre-collision Pose (PVCP) dataset and proposed a simple framework for Pedestrian Pre-collision Pose and Shape Estimation (PPSENet) in collision accident videos. We collected dashcam videos and used existing pose estimation algorithms to obtain rough 2D keypoints and 3D mesh initialization results, followed by manual correction using specialized annotation tools. Specifically, we designed an SMPL annotation tool (25) to align the initial results with image contours, resulting in approximately 40K+ frames of accident images and 20K+ instances of pedestrian emergency poses with both 2D and 3D annotations. Our PPSENet estimates the 2D pose from images (Image to Pose, ITP) and lifts the 2D pose to the 3D mesh (Pose to Mesh, PTM). We used a pre-trained model (12) to capture prior knowledge of human actions and employed iterative regression (26; 11; 27) to estimate pedestrian pre-collision poses and shapes. Additionally, we classified emergency poses and introduced pose class loss, achieving superior accuracy compared to existing methods.

The main contributions of this paper are summarized below:

- We constructed a pedestrian pre-collision pose dataset, PVCP, by collecting dashcam videos of pedestrian-vehicle collisions. Through algorithm initialization and manual annotation, we obtained rich pose representation annotations, including 2D, 3D keypoints and SMPL mesh.

- We propose a two-stage pedestrian pre-collision pose and shape estimation network, PPSENet, which first estimates the 2D pose from the image and then lifts the 2D pose to the 3D pose. A pretrained encoder with pose estimation and an iterative regression decoder are combined, and introduce a collision pose class loss.

- Our framework achieved promising results on the PVCP dataset, outperforming other methods of human pose estimation. This provides both data and algorithmic support for pedestrian pre-collision pose estimation and active safety protection for pedestrians.

## 2 Related Work

**Pedestrian Pre-collision Pose.** Pedestrian pre-collision pose is crucial for studying collision damage, as the initial posture at the time of impact directly affects the severity and nature of the injuries (28; 6; 29; 30; 3; 31). Early studies estimated collision poses by collecting post-accident data from pedestrians and vehicles (32). Cadaver tests (33) became effective for biomechanical damage studies but are limited by ethics, sample size, and high costs. Currently, collision simulation software is the most convenient and effective method to assess damage under various poses (2; 3; 4; 5). However, initial collision poses are often fixed templates or simple categories (6; 2; 3; 4; 5), differing significantly from real pre-collision poses. One method to obtain pre-collision poses is using motion capture in virtual environments with volunteers (34), but this is limited by device constraints and lack of real danger. Another method captures collision sequences from real accident videos, manually measuring posture angles or adjusting dummies to match real collision poses (33). This method is closer to real accident scenarios but is labor-intensive, time-consuming, and lacks standardized testing, limiting its use to single accident reconstructions. With advances in deep learning and computer vision, some research has employed human pose estimation algorithms to automatically extract collision poses from accident images (15), providing a new approach to acquiring pre-collision poses

Table 1: Comparison of datasets on *Accident Warning*, *Traffic Scene* and *Pedestrian Pose*. 'V' represents the vehicle perspective, 'M' represents the monitoring perspective, 'D' represents a dynamic background and 'S' represents a static background.

| Type | Dataset | Year | Perspective | Background | Detection | Track | Depth | Pose | Shape | Class | Frame |
|---|---|---|---|---|---|---|---|---|---|---|---|
| **Accident Warning** | DAD(22) | 2016 | V | D | ✓(2D Bbox) | ✓ | × | × | × | × | >62k |
| | ShanghaiTech(46) | 2017 | M | S | ✓(Mask) | ✓ | × | × | × | × | >300k |
| | A3D(23) | 2019 | V | D | ✓(2D Bbox) | ✓ | × | × | × | × | >128k |
| | DADA(47) | 2019 | V | D | ✓(3D Bbox) | × | × | × | × | × | >650k |
| | CCD(24) | 2020 | V | D | ✓(2D Bbox) | ✓ | × | × | × | × | >75k |
| **Traffic Scene** | KITTI(19) | 2012 | V | D | ✓(3D Bbox) | ✓ | ✓ | × | × | × | >30K |
| | Cityscapes(48) | 2015 | V | D | ✓(Mask) | × | × | × | × | × | >5k |
| | CityPersons(49) | 2016 | V | D | ✓(2D Bbox) | × | × | × | × | × | >5k |
| | MOT(50) | 2012-2017 | V/M | D/S | ✓(2D Bbox) | ✓ | × | × | × | × | – |
| | Nuscenes(18) | 2019 | V | D | ✓(3D Bbox) | ✓ | ✓ | × | × | × | >35k |
| **Pedestrian Pose** | MSCOCO(20) | 2014-2017 | Daily scene | S | ✓(2D Bbox) | × | × | ✓(2D) | × | × | >1000k |
| | Human3.6M(16) | 2014 | M | S | ✓(2D Bbox) | ✓ | ✓ | ✓(2D/3D) | × | × | >500k |
| | PW3D(21) | 2018 | hand-held camera | D | × | ✓ | × | ✓(3D) | × | × | >50k |
| | Accident Video(15) | 2020 | V/M | D/S | × | ✓ | × | × | × | – | |
| | PedX(14) | 2018 | M | S | ✓(Mask) | ✓ | ✓ | ✓(2D/3D) | ✓ | × | >10k |
| **Ours** | **PVCP** | **2024** | **V(Dashcam)** | **D/S** | **✓(2D Bbox)** | **✓** | **✓** | **✓(2D/3D)** | **✓** | **✓** | **>40k** |

in real accidents. Rapid and accurate acquisition of pedestrian pre-collision poses supports research on collision damage and active safety protection.

**Human Pose Estimation.** Human Pose Estimation (HPE) is a fundamental task of computer vision, which aims to obtain human pose information such as joint positions and angle from images and videos (35). It can be simply classified into 2D human pose estimation and 3D human pose estimation. 2D HPE regresses pixel coordinates $(x, y)$ of joints, while 3D HPE includes depth to obtain three-dimensional coordinates $(x, y, z)$ (7; 8; 36). Though 3D coordinates can be regressed directly from images (37; 38; 39; 40; 41), using 2D pose as intermediate supervision before lifting to 3D often achieves higher accuracy (9; 42). Additionally, the SMPL (Skinned Multi-Person Linear Model) (43) has gained popularity for providing pose and morphological information, along with prior knowledge of body structure, avoiding issues with limb length changes (44; 11; 12; 27). This rotation-based model is particularly useful in biomechanical research (15; 45), which benefits the study of pedestrian emergency poses. In our research, we used 2D-to-3D lifting to estimate the pre-collision pose and shape of pedestrians from real accident videos.

**Accident and Pedestrian Datasets.** Collecting 3D pose datasets in complex traffic scenes poses challenges due to the dynamic environments and uncertain pedestrian poses. While existing large-scale datasets focus on 2D poses(20; 51; 52), 3D pose datasets are often confined to indoor settings using Motion Capture (Mocap) systems(16; 17) or estimated via Inertial Measurement Units (IMUs) for outdoor poses(21). Models trained on indoor datasets do not adapt well to other in-the-wild tasks. Advanced pose estimation methods can generate pseudo-datasets to construct 3D pose datasets in the wild. Although pseudo-3D labels from semi-automatic(44; 53) or fully-automated methods(54; 55) are less accurate than Mocap data and may contain noise, they significantly improve regression-based methods(56). Using a semi-automatic method, we collected dashcam videos of collisions to create a pedestrian-vehicle collision pose dataset, offering rich annotation information and contributing to pedestrian protection tasks. Table 1 highlights its advantages over other datasets.

## 3 PVCP Dataset

### 3.1 Data Collection

Dashcams or public surveillance devices are the main sources of crash data (22; 23; 24; 15). Dashcam views dominate vision-based Traffic Accident Anticipation (TAA) datasets due to the high potential for collision avoidance through vehicle control (57). Our PVCP dataset are all derived from the vehicular perspective of dashcam, and videos are sourced from two primary origins. Similar to previous works (47; 22; 23; 24; 15), we collected videos of pedestrian and vehicle collisions from online platforms such as YouTube, using 'pedestrian-vehicle collision' as a keyword. In addition, a small number of videos are derived from existing open-source traffic datasets (47; 22; 23), which were primarily developed for tasks related to driver attention and the prediction of sudden accidents. All of the collected videos were reduced to individual accident footage, recording a complete pedestrian

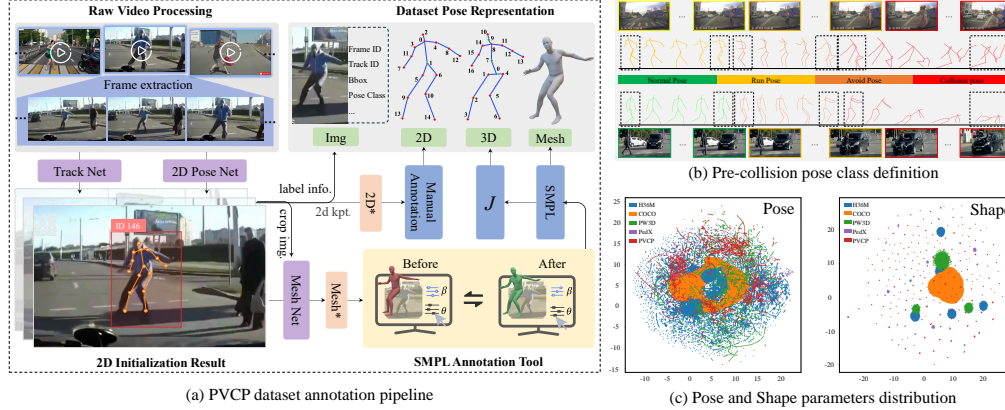

**Figure 1:** (a) PVCP dataset annotation pipeline. (b) Pose class definition. Different colors are used to represent different pose types. (c) Dataset attribute distributions. Utilizing UMAP(58), the pose parameters $\theta \in \mathbb{R}^{N \times 72}$, and shape parameters shape $\beta \in \mathbb{R}^{N \times 10}$ are reduced to a two-dimensional $\mathbb{R}^{N \times 2}$ embedding space (no unit), with coordinates along the $x$ and $y$-axes, respectively.

and vehicle collision, resulting in 209 pedestrian-vehicle accident videos, totaling 42,511 frames of images and about 19,533 pre-collision poses.

## 3.2  Data Annotations

For the application of pedestrian-vehicle collision accident reconstruction and vehicle active protection in traffic collision scenarios, PVCP provides a rich pedestrian pose representation, including pedestrian clipping images, pedestrian Bounding Box, track id, 2D and 3D keypoints and SMPL mesh label. The entire annotation pipeline is shown in Figure 1(a).

**Pedestrian Pre-collision Pose.** Manually annotating human keypoints is tedious and labor-intensive. We address this by initializing the annotation process with pose estimation models and refining the results manually. First, we identify collision-involved pedestrians in each video using a tracking network (59) and manual filtering, followed by pose annotation. We use a 15 keypoint representation similar to the JHMDB dataset (60) for efficient pose depiction. ViTPose (10) provides rough 2D pose annotations, which we manually adjust for accuracy. For occluded limbs, we estimate positions to complete the 2D skeleton, excluding body parts beyond the frame. Annotating 3D human poses in-the-wild remains challenging with images as the only source. Unlike Pseudo-GT annotators (61; 62), we use SPIN (11) to initialize predictions from cropped images, then refine the SMPL model parameters $\theta \in \mathbb{R}^{24 \times 3}$ and $\beta \in \mathbb{R}^{10}$ using our specially designed SMPL annotation tool (25) for better pixel alignment. This yields mesh pose annotations $\mathcal{M}(\theta, \beta) \in \mathbb{R}^{6890 \times 3}$ for pre-collision pedestrians. Finally, we apply the pre-defined joint regression matrix $\mathcal{J} \in \mathbb{R}^{J \times 6890}$ (44) to obtain 3D keypoints $X_{3D} \in \mathbb{R}^{J \times 3}$ from $\mathcal{J}(\mathcal{M}(\theta, \beta))$, where $J = 17$ (16), as shown in Figure 1(a).

**Pedestrian Motion Class.** Throughout the course of a pedestrian-vehicle collision event, pedestrians often undergo a series of rapid evasive action changes in a short period. Effectively and accurately distinguishing and predicting these imminent changes is crucial for the proactive safety features of vehicle driving systems. Through the observation and analysis of all collected accident videos, as shown in Figure 1(b), we categorize pedestrian behaviors in collision sequences into four types:

*Normal pose*: Represents the pedestrian's pose under normal, non-emergency conditions. This includes upright body pose and natural stances, reflecting the general behavior of pedestrians when not faced with emergencies.

*Run pose:* Characterized by the pedestrian's body leaning forward with rapid alternation of arms and legs, this pose is an active measure to prevent vehicle collisions. It reflects a preemptive action to swiftly move away from potential threats, serving as a strategic pose to avoid accidents.

*Avoid pose:* This pose is adopted by pedestrians upon detecting an imminent collision or other emergency situations. It includes potential actions such as jumping, quickly turning around, and dodging, reflecting the emergency response of pedestrians when recognizing potential danger.

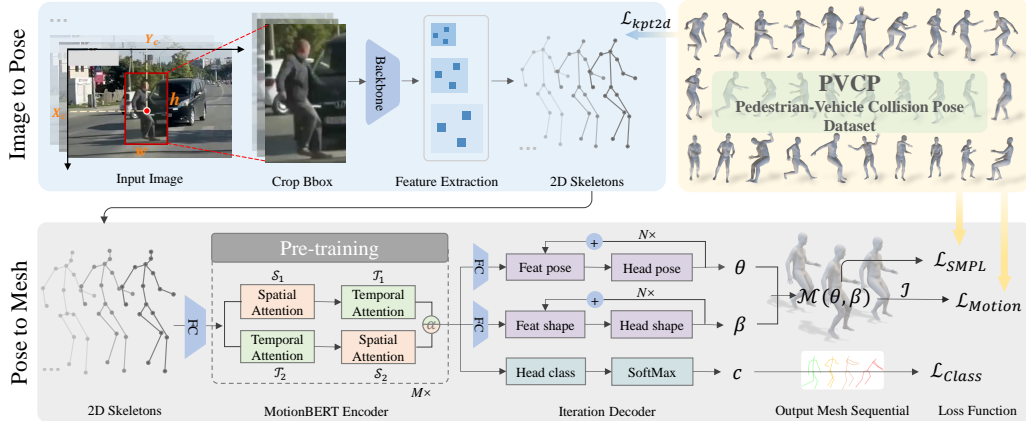

Figure 2: The overview of the PPSENet. It consists of two stages: image to 2D pose (ITP) and 2D pose to 3D Mesh (PTM).

***Collision pose:*** Represents the pose of a pedestrian post-collision with a vehicle. It encompasses possible actions such as losing balance, falling, and sustaining injuries, reflecting the change in the physical state of pedestrians after a collision.

The sequences of two behavior annotations are illustrated in Figure 1(b). It is important to note that a complete collision incident does not necessarily encompass all four types of behavior, nor is there a fixed sequence. Because some accidents do not result in a final collision due to timely measures taken, and some pedestrians may not even be aware of the approaching vehicle. To distinguish between the four types, we employ a four-digit one-hot encoding for pose annotation. For some critical changes in pose, we use three annotators and take the majority's annotation result as the final Ground Truth.

### 3.3 Dataset attribute distributions

PVCP dataset contains the pose sequences during pedestrian collision. We visualized the pose and shape distribution of SMPL labels in PVCP dataset and compared them with those in commonly used and influential datasets in human pose estimation, including an indoor MoCap pose dataset Human3.6M (16), a in-the-wild dataset PW3D (21), a pseudo-3D labels dataset MSCOCO (20) and a traffic scenario pedestrian dataset PedX (14) are shown in the Figure 1(c). The distribution of pose parameters in our dataset is comparable to PW3D and significantly larger than that of everyday poses in MSCOCO. Our dataset contains over 200 individual collision pedestrians, and its shape parameter distribution is more extensive than other datasets that use a limited number of actors, such as 5 actors in the Human3.6M dataset. The most similar to PVCP is PedX, but its distribution is sparse due to its small number of poses. To sum up, our PVCP has good generalization in pose and shape parameters, representing pedestrian pose under a variety of collision conditions. Compared to existing pose datasets, PVCP demonstrates significant differences and advantages in terms of scene specificity, action space variation, and temporal continuity.

## 4 Network Architecture

In this section we present a pipeline for pedestrian Pre-collision Pose and Shape Estimation (PPSE), as shown in Figure 2, adopts a top-down two-stage strategy. Instead of estimating the pose of all pedestrians in the whole images, we pre-select the Bbox of collision pedestrians obtained by the detect and track networks (59). The crop of a single pedestrian was input into the Image to Pose (ITP) network to extract the image features of the collision pedestrian and estimate the pedestrian's 2D pose. Then, the 2D pose was lifted to 3D mesh through the Pose to Mesh (PTM) network.

### 4.1 Pedestrian Pre-collision Pose and Shape Estimation

**Image to Pose**. Estimating 2D human pose from images is a basic and mature task, and many works have achieved very effective results in different datasets. We take image frame $I$ and corresponding

*Bbox* as input, simply use ResNet50 (63) as the backbone of feature extraction, and use a transposed convolution and a heat map regression head $\mathcal{H}$ as the 2D pose estimation network, which is also the classic paradigm of 2D pose estimation (8).

**Pose to Mesh**. PTM is a network architecture for 2D-to-3D lifting, we take the 2D pose sequence $P_{2D}^{L} \in \mathbb{R}^{T \times J \times C_{in}}$ as input. First, a Dual-stream Spatio-temporal Transformer (DSTformer) is used as an encoder to extract the spatio-temporal features of the pose sequence $F_m \in \mathbb{R}^{T \times J \times C_f}$. Then we designed an iterative regression decoder to obtain pose parameters and shape parameters respectively. At the same time, we added an additional regression head of pose class to coordinate with pose class loss to further improve the precision of pose regression. Where $T$ represents the length of the pose sequence and $J$ represents the number of pedestrian joints. $J = 17$ takes the Human3.6M dataset (16) joint format and we generate additional nodes from $P_{2d} \in \mathbb{R}^{15 \times 2}$.

Inspired (12), PTM uses the backbone of an pre-trained model, MotionBERT (12), and combines PVCP dataset to estimate pedestrian pre-collision pose in collision scenarios. MotionBERT is a pre-training model of human motion representations. Firstly, it learns the prior knowledge of human motion poses under the training of a large number of datasets, which is suitable for the further improvement of small datasets such as PVCP and special pose types such as pre-collision poses. Secondly, its training mode simulates the detection results by randomly masking and adding noise to 2D pose sequences. In the same way, we train the situation of vehicles shielding pedestrian's lower limbs in collision environment. This encoder stacks spatial and temporal Multi-Head Self-Attention (MHSA) blocks in different orders to form two parallel computation branches:

$$F^i = \alpha_{ST}^i \circ \mathcal{T}_1^i(\mathcal{S}_1^i(F^{i-1})) + \alpha_{TS}^i \circ \mathcal{S}_2^i(\mathcal{T}_2^i(F^{i-1})) \tag{1}$$

$$\alpha_{ST}^i, \alpha_{TS}^i = softmax(\mathcal{W}_f(\mathcal{T}_1^i(\mathcal{S}_1^i(F^{i-1})) \oplus \mathcal{S}_2^i(\mathcal{T}_2^i(F^{i-1})))) \tag{2}$$

Where $i \in 1, \ldots, M$ and $M$ represents network depth. $S_i$ and $T_i$ represent Spatial MHSA and temporal MHSA of different depth layers, respectively. Adaptive fusion weights $\alpha_{ST}, \alpha_{TS}$ fuses the output features of the two branches using adaptive weights predicted by an attention regressor. $\mathcal{W}_f$ denotes linear layer, $\oplus$ denotes concatenation.

We divide the fusion feature $F_m \in \mathbb{R}^{T \times J \times C_f}$ obtained by DSTFormer into three branches using different linear layers. That is, pose feature $F_\theta \in \mathbb{R}^{T \times C_h}$, shape features $F_\beta \in \mathbb{R}^{T \times C_h}$ and class features $F_c \in \mathbb{R}^{T \times C_h}$. Then the three features are predicted by three different heads, outputting the pose parameter $\theta \in \mathbb{R}^{T \times 24 \times 6}$, shape parameter $\beta \in \mathbb{R}^{10}$ and class probability $c \in \mathbb{R}^{T \times 4}$. Here, we use the 6D rotation representation to converge the pose parameter more quickly (64). Further, we adopted the idea of iterative regression to add the fusion features of pose and form with the predicted results and iteratively output:

$$\theta^k = W_\theta^k(F_\theta) + \theta^{k-1}; \quad \beta^k = W_\beta^k(F_\beta) + \beta^{k-1}; \quad c = softmax(W_c(F_c)) \tag{3}$$

Where $k \in 1, \ldots, N$ and $N$ represents the number of iterations. $W_\theta$, $W_\beta$ and $W_c$ are three linear transformations heads, $\oplus$ denotes concatenation, $+$ denotes add.

### 4.2 Loss Function

The loss function of ITP, defined as the Mean Squared Error (MSE), is applied for comparing the predicted heatmaps $\hat{H}$ and the ground truth heatmaps $H$. The heatmap for joint $k$ is generated by applying a 2D Gaussian centered on the $k_{th}$ joint's location.

$$\mathcal{L}_{ITP} = \left\| \hat{H} - H \right\|_2 \tag{4}$$

The PTM loss function consists of three parts: SMPL loss, motion loss, and the pose class loss introduced by us. The final loss function is calculated as:

$$\mathcal{L}_{PTM} = \mathcal{L}_{SMPL} + \mathcal{L}_{Motion} + \mathcal{L}_{Class} \tag{5}$$

**SMPL loss:** The loss function of SMPL-based 3D human mesh usually consists of three parts:

$$\mathcal{L}_{SMPL} = \lambda_\theta \mathcal{L}_\theta + \lambda_\beta \mathcal{L}_\beta + \lambda_n \mathcal{L}_{norm} \tag{6}$$

Where $\mathcal{L}_\theta = \left\|\hat{\theta} - \theta\right\|_1$, $\mathcal{L}_\beta = \left\|\hat{\beta} - \beta\right\|_1$, $\mathcal{L}_{norm} = \left\|\hat{\theta}\right\|_2 + \left\|\hat{\beta}\right\|_2$ represents pose loss, shape loss and normalization loss respectively.

**Motion Loss:** The human body is a complex rigid structure interconnected between joints, and the continuous frame sequence of human body movements possesses certain temporal characteristics. Therefore, introducing motion loss as

$$\mathcal{L}_{Motion} = \lambda_k \mathcal{L}_{kp3D} + \lambda_v \mathcal{L}_v \qquad (7)$$

Where $\mathcal{L}_{kp3D} = \left\|\hat{X} - X\right\|_1$ represents the loss of 3D keypoints. $\mathcal{L}_v = \left\|\hat{V} - V\right\|_1$ represents speed loss, and $V = X_{t+1} - X_t$, $\hat{V} = \hat{X}_{t+1} - \hat{X}_t$.

**Pose class loss:** The pedestrian pose of the collision sequence has obvious categories, namely normal pose, running pose, avoiding pose and collision pose as described in Sec. 3.2, the class loss of pose is defined as follows:

$$\mathcal{L}_{Class} = \lambda_c \mathcal{L}_{Cross\ Entropy}(\hat{C}, C) \qquad (8)$$

where $\mathcal{L}_{Cross\ Entropy}(\hat{C}, C)$ epresents the cross entropy loss between the predicted pose class and the GT pose class. $\lambda_\theta, \lambda_\beta, \lambda_n, \lambda_k, \lambda_v, \lambda_c$ are the constants of the balance weight loss.

# 5 Experimental and Results

## 5.1 Dataset, Evaluation Metric and Implementation Details

**Dataset.** The PVCP dataset consists of over 20K+ pedestrian pre-collision poses, with 19,533 poses annotated with category labels. Subsequently, we selected 164 video sequences as the trainset and 45 video sequences as the testset. To ensure the effectiveness of the pose sequences, only poses with the number of keypoints $N_{kpt} \geq 10$ were selected, resulting in 15,458 poses for training (*Normal*:7,912; *Run*:5,044; *Avoid*:2,289; *Collision*:213) and 5,503 poses for testing (*Normal*:3,383; *Run*:1,431; *Avoid*:631; *Collision*:58). The entire training process only utilized the PVCP dataset, while the pre-training model weights were obtained from MotionBERT (12) trained on the AMASS (17), Human3.6M (16), PW3D (21), and MSCOCO (20) datasets.

**Evaluation Metric.** We evaluated the estimation of 3D human pose and shape using the following metrics: MPJPE($mm, \downarrow$), PA-MPJPE($mm, \downarrow$), MPVE($mm, \downarrow$), PA-MPVE($mm, \downarrow$). Further, we test the errors of 14 keypoints ($X\_14j$) shared by 2D and 3D pose representations (21) and 17 keypoints ($X\_17j$) represented only by 3D pose representations (16) respectively.

**Implementation Details.** PyTorch (65) was used for the entire experimental environment, four NVIDIA RTX 2080Ti GPUs for all training, and batchsize was uniformly set to 32. In the training stage, we only use the images from the PVCP trainset and the corresponding 2D ground truth keypoints as the input of the two models. We first train the ITP network by loading a pre-trained model of the MPII dataset (51) and training 40 epochs. For PTM, we use sequence length $T = 16$ and train 100 epochs in about two hours. In the test stage, in addition to the 2D ground truth keypoints of the testset, we also take the image of the testset as input to the ITP model, and then take the estimated 2D keypoint results as input to the PTM model. The effects of the same PTM training model with two different inputs are compared.

## 5.2 Effects of Dataset and Pose priors

We first evaluate the effects of PVCP dataset on improving the pedestrians pre-collision pose estimation. Compared to large-scale human pose datasets, our PVCP dataset is not numerically dominant, so we use a pre-trained model that learns human pose priors and fine-tune it based on that. We ran tests on the original MotionBERT (12) to compare scratch training and the PVCP trainset with together pre-trained models. We take the detected 2D pose sequence (2D Det) as input, and compare the errors of four pre-collision pose (Normal, Run, Avoid, Collision) and all pose (All). As shown in the Table 2, the results of only-pretrain model or only-PVCP trainset are relatively poor. Due to the difference between the pre-collision pose and the daily pose, the error of using only-pretrain model ($MPVE_{det} = 335.11mm$) is even worse than that of only-PVCP dataset ($MPVE_{det} = 315.64mm$). When the pre-trained model and the PVCP dataset are trained together, The minimum error ($MPVE_{det} = 282.50mm$) is obtained.

Table 2: Effects of Dataset and Pre-training. Top use detected 2D pose sequences. Bottom use GT 2D pose sequences.

| Input | Train Set | testset | Pose class | MPVE | PAMPVE | MPJPE_14j | PAMPJPE_14j | MPJPE_17j | PAMPJPE_17j |
|---|---|---|---|---|---|---|---|---|---|
| 2D Det | PVCP | PVCP | Normal | 315.94 | 160.25 | 272.18 | 130.72 | 246.42 | 121.30 |
| | | | Run | 318.29 | 189.84 | 274.78 | 160.35 | 246.95 | 145.07 |
| | | | Avoid | 305.01 | 159.19 | 260.31 | 121.42 | 232.56 | 113.21 |
| | | | Collision | 347.53 | 171.82 | 311.88 | 145.64 | 281.35 | 139.46 |
| | | | All | 315.64 | 168.11 | 271.91 | 137.75 | 245.35 | 126.92 |
| | Pretrain | PVCP | Normal | 347.10 | 190.17 | 312.21 | 154.85 | 285.62 | 145.55 |
| | | | Run | 309.19 | 183.27 | 277.01 | 152.19 | 251.53 | 141.11 |
| | | | Avoid | 330.18 | 189.69 | 293.76 | 155.54 | 264.89 | 144.38 |
| | | | Collision | 334.14 | 164.32 | 301.52 | 133.26 | 275.28 | 128.19 |
| | | | All | 335.11 | 188.09 | 300.80 | 154.06 | 274.27 | 144.12 |
| | Pretrain + PVCP | PVCP | Normal | 294.73 | 170.10 | 253.80 | 137.39 | 232.74 | 128.24 |
| | | | Run | 253.16 | 149.99 | 219.06 | 124.01 | 200.19 | 115.27 |
| | | | Avoid | 286.85 | 159.69 | 246.94 | 124.86 | 222.02 | 114.96 |
| | | | Collision | 250.58 | 161.25 | 222.47 | 127.37 | 200.38 | 120.47 |
| | | | All | **282.50** | **163.58** | **243.59** | **132.43** | **222.70** | **123.33** |
| 2D GT | PVCP | PVCP | Normal | 304.65 | 167.56 | 260.68 | 138.49 | 233.70 | 126.83 |
| | | | Run | 296.75 | 192.00 | 254.58 | 163.80 | 226.49 | 146.66 |
| | | | Avoid | 277.51 | 157.48 | 234.30 | 123.02 | 206.55 | 113.44 |
| | | | Collision | 354.76 | 178.22 | 319.56 | 154.95 | 287.38 | 146.83 |
| | | | All | 300.04 | 173.09 | 256.69 | 143.73 | 229.30 | 130.85 |
| | Pretrain | PVCP | Normal | 175.24 | 111.72 | 152.10 | 87.68 | 138.87 | 82.11 |
| | | | Run | 153.45 | 107.26 | 131.93 | 84.02 | 118.99 | 77.76 |
| | | | Avoid | 143.32 | 93.61 | 122.91 | 73.45 | 111.33 | 68.89 |
| | | | Collision | 151.18 | 91.20 | 133.60 | 77.66 | 124.71 | 71.06 |
| | | | All | 165.90 | 108.48 | 143.52 | 85.14 | 130.56 | 79.48 |
| | Pretrain + PVCP | PVCP | Normal | 156.06 | 103.16 | 132.74 | 80.59 | 120.35 | 74.92 |
| | | | Run | 129.49 | 89.31 | 109.93 | 70.91 | 100.19 | 65.70 |
| | | | Avoid | 127.04 | 85.36 | 108.30 | 65.35 | 96.74 | 60.44 |
| | | | Collision | 135.89 | 89.71 | 127.11 | 70.86 | 112.50 | 64.94 |
| | | | All | **145.77** | **97.50** | **124.04** | **76.34** | **112.43** | **70.87** |

Table 3: Component of system. Top use detected 2D pose sequences. Bottom use GT 2D pose sequences.

| Input | Pretrain | Iter | Class Loss | Pose class | MPVE | PAMPVE | MPJPE_14j | PAMPJPE_14j | MPJPE_17j | PAMPJPE_17j |
|---|---|---|---|---|---|---|---|---|---|---|
| 2D Det | ✓ | | | All | 282.50 | 163.58 | 243.59 | 132.43 | 222.70 | 123.33 |
| | ✓ | 3 | | All | 266.20 | 146.88 | 225.38 | 116.99 | 204.98 | 108.63 |
| | ✓ | | ✓ | All | 259.05 | **143.52** | 220.39 | 115.47 | 200.16 | 107.03 |
| | ✓ | 3 | ✓ | All | **257.75** | 144.19 | **218.61** | **114.50** | **198.16** | **105.86** |
| 2D GT | ✓ | | | All | 145.77 | 97.50 | 124.04 | 76.34 | 112.43 | 70.87 |
| | ✓ | 3 | | All | 145.75 | 96.69 | 123.16 | 75.13 | 111.90 | 69.89 |
| | ✓ | | ✓ | All | 141.28 | **92.78** | 120.16 | **72.43** | 108.90 | **67.58** |
| | ✓ | 3 | ✓ | All | **140.43** | 96.43 | **118.80** | 75.13 | **107.47** | 69.56 |

When 2D ground truth pose sequence (2D GT) is used as the input, the result without using the pre-trained model is relatively poor ($MPVE_{gt} = 300.04mm$), and the error after using only-pretrain model is significantly decreased ($MPVE_{gt} = 165.90mm$), because the number of keypoints ($\geq 10$) of the 2D GT pose sequences are relatively complete compared with the 2D Det pose sequences, its input is not affected by lighting conditions, background appearance, clothing, and weather conditions. Similarly, when the pre-trained model and the PVCP dataset are trained together, The minimum error ($MPVE_{gt} = 145.77mm$) is obtained. This shows that our PVCP dataset has different features from ordinary pose, and the pose prior of the pre-trained model can effectively promote the precision of the pre-collision pose.

## 5.3 Ablation Study

In the stage of PTM, we added an iterative decode, which does not directly predict the output once but gradually approximates the optimal solution with multiple iterations. At the same time, an pose classification regression head is added to use class loss as supervision. As shown in Table 3, we compared the impact of different components on network performance, loaded the pre-trained model each time, and set the optimal number of iterations to 3 (as shown in Table 4).

Table 4: Comparison of 2D GT input in different iterations number.

| Iter | Pose class | MPVE | PAMPVE | MPJPE_14j | PAMPJPE_14j | MPJPE_17j | PAMPJPE_17j |
|------|-----------|------|--------|-----------|-------------|-----------|-------------|
| 2 | All | 141.95 | 97.43 | 120.04 | 75.45 | 108.63 | 69.85 |
| 3 | All | 140.43 | **96.43** | 118.80 | **75.13** | 107.47 | **69.56** |
| 4 | All | **139.96** | 96.92 | **118.46** | 75.19 | **107.16** | 69.62 |
| 5 | All | 140.01 | 97.10 | 118.54 | 75.40 | 107.27 | 69.83 |
| 6 | All | 140.41 | 97.42 | 118.89 | 75.70 | 107.68 | 70.14 |

Table 5: Comparison of state-of-the-art methods on the PVCP testset. [†] denotes that the training weights provided by the official are used, and * denotes the model weights trained together with the PVCP trainset.

| Paradigm | Method | Pose class | MPVE | PAMPVE | MPJPE_14j | PAMPJPE_14j | MPJPE_17j | PAMPJPE_17j |
|----------|--------|-----------|------|--------|-----------|-------------|-----------|-------------|
| One Stage | [†]VIBE(66) | Normal | 856.87 | 234.47 | 731.90 | 217.35 | – | – |
| | | Run | 856.10 | 232.67 | 732.33 | 226.45 | – | – |
| | | Avoid | 777.92 | 227.16 | 664.25 | 216.72 | – | – |
| | | Collision | 950.47 | 212.21 | 869.86 | 202.01 | – | – |
| | | All | 849.09 | 233.08 | 725.92 | 219.55 | – | – |
| | [†]PARE(67) | Normal | 225.99 | 147.04 | 193.62 | 114.35 | – | – |
| | | Run | 235.99 | 180.98 | 193.40 | 137.08 | – | – |
| | | Avoid | 210.02 | 143.88 | 176.76 | 109.10 | – | – |
| | | Collision | 247.18 | 167.62 | 225.96 | 132.89 | – | – |
| | | All | **226.98** | 155.72 | **191.97** | 119.85 | – | – |
| Two Stage | [†]Pose2Mesh(68) | Normal | 247.24 | 148.87 | 222.34 | 122.42 | – | – |
| | | Run | 255.26 | 181.16 | 222.33 | 145.14 | – | – |
| | | Avoid | 217.97 | 141.43 | 191.38 | 112.35 | – | – |
| | | Collision | 231.65 | 174.44 | 210.44 | 145.54 | – | – |
| | | All | 245.88 | 156.69 | 218.71 | 127.41 | – | – |
| | *MotionBERT(12) | Normal | 294.73 | 170.10 | 253.80 | 137.39 | 232.74 | 128.24 |
| | | Run | 253.16 | 149.99 | 219.06 | 124.01 | 200.19 | 115.27 |
| | | Avoid | 286.85 | 159.69 | 246.94 | 124.86 | 222.02 | 114.96 |
| | | Collision | 250.58 | 161.25 | 222.47 | 127.37 | 200.38 | 120.47 |
| | | All | 282.50 | 163.58 | 243.59 | 132.43 | 222.70 | 123.33 |
| | *PPSE(Ours) | Normal | 272.79 | 149.02 | 230.49 | 117.47 | 209.99 | 109.04 |
| | | Run | 226.22 | 133.45 | 193.75 | 109.50 | 174.47 | 100.73 |
| | | Avoid | 251.60 | 143.52 | 212.75 | 109.75 | 190.00 | 100.09 |
| | | Collision | 217.68 | 134.95 | 201.15 | 113.10 | 174.57 | 105.94 |
| | | All | 257.75 | **144.19** | 218.61 | **114.50** | **198.16** | **105.86** |

Due to the difference between 2D Det pose sequences and 2D GT pose sequences in the number and correct position of keypoints, when only iterative decode is used, the input error of 2D Det decreases significantly ($MPVE_{det} = 282.50mm \rightarrow 266.20mm$), while that of 2D GT decreases slightly ($MPVE_{gt} = 145.77mm \rightarrow 145.75mm$), which may be because multiple iterations improve the pose regression ability of incomplete pose. When only pose class loss is added, the error reduction space of the 2D GT input is significantly stronger ($MPVE_{gt} = 145.75mm \rightarrow 141.28mm$) than that of the 2D Det input ($MPVE_{det} = 266.20mm \rightarrow 259.05mm$), possibly because the pose that is complete and correctly positioned at the keypoints has a stronger correlation with the pose class label. Under the combined action of iterative regression decoder and loss function, both 2D pose sequence inputs achieve the minimum error ($MPVE_{det} = 257.75mm, MPVE_{gt} = 140.43mm$).

## 5.4 Comparison with the state-of-the-art

**Quantitative comparison.** Similar to (69), Table 5 reports results for multiple baselines on the PVCP testset using the evaluation metrics described in Sec. 5.1. We compare the classic baseline methods of two paradigms: the one-stage method, which involves direct regression from image to mesh, and the two-stage method, which involves regression from 2D pose to 3D mesh. Following (66; 67), in the one-stage method, we used ResNet-50 (63) to extract the feature $f^i \in \mathbb{R}^{2048}$ of the collision pedestrian clip-off image in each frame. Following (68), We use DarkPose (70) to extract 2D poses in COCO format (20). For MotionBERT (12) and our method, 2D pose is converted from 15 keypoints of JHMDB (60) to the corresponding 17 keypoints of Human3.6 (16) as input. Because the PVCP dataset contains only pose and shape annotations, there is a lack of spatial arrangement in the 3D scene. Therefore, the error of MPVE and MPJPE is large, but in PAMPVE and PAMPJPE, our method achieves the best accuracy. In addition, in the one-stage method, the effect of PARE (67) is relatively excellent, because PARE has optimized the occlusion of pedestrians and is well adapted

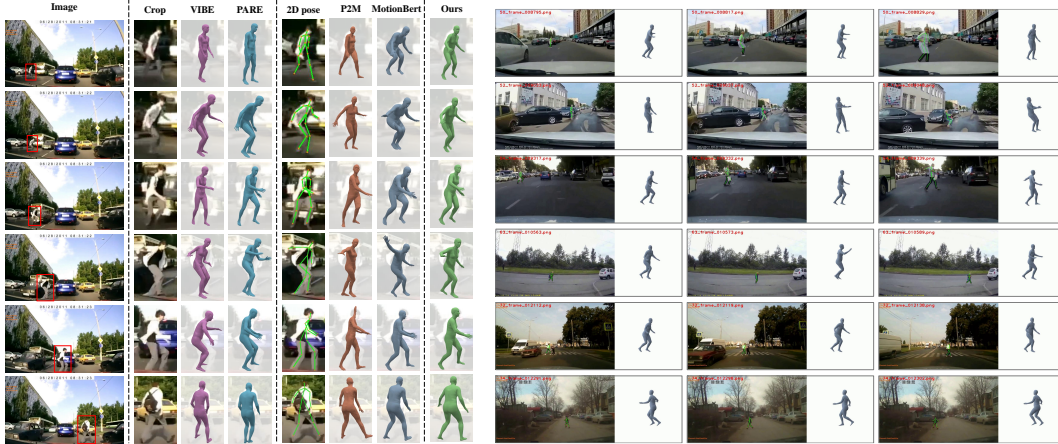

Figure 3: Qualitative comparison. Left: Comparison with SOTA methods in PVCP testset. VIBE (66) and PARE (67) take images as input, P2M (Pose2Mesh) (68) and MotionBERT(12) take detected 2D pose as input. Right: Output examples of our method in PVCP testset.

to the real situation of the occlusion of pedestrians' lower limbs in the collision scene. In addition, the two-stage method needs to detect the 2D pose first, and the missing keypoints detection caused by occlusion will continue the error to the subsequent Mesh regression stage. However, in the case of single-class pre-collision poses (*Run, Avoid, Collision*), our method still has great advantages.

**Qualitative comparison.** A qualitative comparison of different methods is shown in Figure 3. Ignoring the global orientation and position, we manually adjust the Mesh output of different methods to the outline of the pedestrian in the picture, focusing on comparing the pre-collision pose itself. It can be seen that our method is most close to the real pedestrian pose in the image, and at the same time close to the estimated 2D pose, especially for the most complex pedestrian upper limb (row 4).

# 6 Conclusion

In this work, we construct the first Pedestrian-Vehicle Collision Pose (PVCP) dataset from the perspective of dashcams, which contains a variety of pedestrian pose representation annotation. At the same time, we propose a framework called PPSENet for the estimation of pedestrian pre-collision pose and shape. Specifically, a two-stage method is adopted: first, the pedestrian's 2D pose is estimated from the image, and then the pedestrian's 3D mesh is estimated from the 2D pose. Furthermore, we adopt the pose prior of the pre-trained model and the idea of iterative regression, introducing the pose class loss to achieve the minimum error on the PVCP testset, effectively estimating the pedestrian pre-collision pose in traffic collision scenarios. We hope that this work will offer new insights into human pose estimation and active pedestrian safety protection.

**Limitations.** Due to the difficulty of collecting the dataset, the dataset is small in size and lacks real camera parameters, vehicle speed information, global position and direction of pedestrians. Additionally, our task involves two-stage of pose estimation, and the final pose error is largely influenced by the 2D pose estimation results from the first stage. Our method is not real-time at present, because our input is Image and pre-selected Bbox sequence of collision pedestrian targets.

**Future Work.** From the comparison with state-of-the-art (SOTA) methods, it is evident that methods like PARE (67), which directly estimate 3D meshes of pedestrians from images, can also achieve good results. Future work can focus on new improvements in one-stage methods. Furthermore, we hope that the introduction of more modal information can further improve the accuracy of the estimation. The purpose of our work is to provide pose data support for the study of vehicle active and passive protection system, so as to facilitate subsequent accident reconstruction, pedestrian injury assessment and vehicle structural design.

## Acknowledgement

This work was supported in part by National Key Research and Development Program of China (Grant No. 2018YFE0192900), and by the Open Funds of State Key Laboratory of Vehicle NVH and Safety Technology (Grant No. NVHSKL-202107).

## References

[1] J. Elliott, C. Simms, and D. Wood, "Pedestrian head translation, rotation and impact velocity: The influence of vehicle speed, pedestrian speed and pedestrian gait," *Accident Analysis Prevention*, vol. 45, pp. 342–353, 2012.

[2] L. Shi, Y. Han, H. Huang, Q. Li, B. Wang, and K. Mizuno, "Analysis of pedestrian-to-ground impact injury risk in vehicle-to-pedestrian collisions based on rotation angles," *Journal of Safety Research*, vol. 64, pp. 37–47, 2018. [Online]. Available: https://www.sciencedirect.com/science/article/pii/S0022437517303705

[3] T. Zou, A. Zha, Q. Liu, and C. Simms, "Pedestrian gaits observed from actual pedestrian-vehicle collisions," *International Journal of Crashworthiness*, vol. 27, no. 1, pp. 1–23, 2022.

[4] S. Shang, C. Masson, M. Llari, M. Py, Q. Ferrand, P.-J. Arnoux, and C. Simms, "The predictive capacity of the madymo ellipsoid pedestrian model for pedestrian ground contact kinematics and injury evaluation," *Accident Analysis Prevention*, vol. 149, p. 105803, 2021.

[5] G. Crocetta, S. Piantini, M. Pierini, and C. Simms, "The influence of vehicle front-end design on pedestrian ground impact," *Accident Analysis and Prevention*, vol. 79, no. Jun., pp. 56–69, 1 2015.

[6] C. D. Untaroiu, M. U. Meissner, J. R. Crandall, Y. Takahashi, M. Okamoto, and O. Ito, "Crash reconstruction of pedestrian accidents using optimization techniques," *International Journal of Impact Engineering*, vol. 36, no. 2, pp. 210–219, 2009.

[7] Z. Cao, T. Simon, S.-E. Wei, and Y. Sheikh, "Realtime multi-person 2d pose estimation using part affinity fields," in *2017 IEEE Conference on Computer Vision and Pattern Recognition (CVPR)*, Jul 2017. [Online]. Available: http://dx.doi.org/10.1109/cvpr.2017.143

[8] B. Xiao, H. Wu, and Y. Wei, "Simple baselines for human pose estimation and tracking," in *Proceedings of the European conference on computer vision (ECCV)*, 2018, pp. 466–481.

[9] J. Martinez, R. Hossain, J. Romero, and J. J. Little, "A simple yet effective baseline for 3d human pose estimation," in *2017 IEEE International Conference on Computer Vision (ICCV)*, Oct 2017. [Online]. Available: http://dx.doi.org/10.1109/iccv.2017.288

[10] Y. Xu, J. Zhang, Q. Zhang, and D. Tao, "Vitpose: Simple vision transformer baselines for human pose estimation," *Advances in Neural Information Processing Systems*, vol. 35, pp. 38 571–38 584, 2022.

[11] N. Kolotouros, G. Pavlakos, M. Black, and K. Daniilidis, "Learning to reconstruct 3d human pose and shape via model-fitting in the loop," in *2019 IEEE/CVF International Conference on Computer Vision (ICCV)*, Oct 2019. [Online]. Available: http://dx.doi.org/10.1109/iccv.2019.00234

[12] W. Zhu, X. Ma, Z. Liu, L. Liu, W. Wu, and Y. Wang, "Motionbert: A unified perspective on learning human motion representations," in *Proceedings of the IEEE/CVF International Conference on Computer Vision*, 2023, pp. 15 085–15 099.

[13] Z. Qiu, Q. Yang, J. Wang, H. Feng, J. Han, E. Ding, C. Xu, D. Fu, and J. Wang, "Psvt: End-to-end multi-person 3d pose and shape estimation with progressive video transformers," in *Proceedings of the IEEE/CVF Conference on Computer Vision and Pattern Recognition*, 2023, pp. 21 254–21 263.

[14] W. Kim, M. S. Ramanagopal, C. Barto, M.-Y. Yu, K. Rosaen, N. Goumas, R. Vasudevan, and M. Johnson-Roberson, "Pedx: Benchmark dataset for metric 3-d pose estimation of pedestrians in complex urban intersections," *IEEE Robotics and Automation Letters*, vol. 4, no. 2, pp. 1940–1947, 2019.

[15] M. Schachner, B. Schneider, W. Sinz, and C. Klug, "Extracting quantitative descriptions of pedestrian pre-crash postures from real-world accident videos," *International Research Council on Biomechanics of Injury (IRCOBI)*, 2020.

[16] C. Ionescu, D. Papava, V. Olaru, and C. Sminchisescu, "Human3. 6m: Large scale datasets and predictive methods for 3d human sensing in natural environments," *IEEE transactions on pattern analysis and machine intelligence*, vol. 36, no. 7, pp. 1325–1339, 2013.

[17] N. Mahmood, N. Ghorbani, N. F. Troje, G. Pons-Moll, and M. Black, "Amass: Archive of motion capture as surface shapes," in *2019 IEEE/CVF International Conference on Computer Vision (ICCV)*, Oct 2019. [Online]. Available: http://dx.doi.org/10.1109/iccv.2019.00554

[18] H. Caesar, V. Bankiti, A. H. Lang, S. Vora, V. E. Liong, Q. Xu, A. Krishnan, Y. Pan, G. Baldan, and O. Beijbom, "nuscenes: A multimodal dataset for autonomous driving," in *Proceedings of the IEEE/CVF conference on computer vision and pattern recognition*, 2020, pp. 11 621–11 631.

[19] A. Geiger, P. Lenz, C. Stiller, and R. Urtasun, "Vision meets robotics: The kitti dataset," *The International Journal of Robotics Research*, vol. 32, no. 11, pp. 1231–1237, 2013.

[20] T.-Y. Lin, M. Maire, S. Belongie, J. Hays, P. Perona, D. Ramanan, P. Dollár, and C. L. Zitnick, "Microsoft coco: Common objects in context," in *Computer Vision–ECCV 2014: 13th European Conference, Zurich, Switzerland, September 6-12, 2014, Proceedings, Part V 13*. Springer, 2014, pp. 740–755.

[21] T. Von Marcard, R. Henschel, M. J. Black, B. Rosenhahn, and G. Pons-Moll, "Recovering accurate 3d human pose in the wild using imus and a moving camera," in *Proceedings of the European conference on computer vision (ECCV)*, 2018, pp. 601–617.

[22] F.-H. Chan, Y.-T. Chen, Y. Xiang, and M. Sun, "Anticipating accidents in dashcam videos," in *COMPUTER VISION - ACCV 2016, PT IV*, ser. Lecture Notes in Computer Science, S. Lai, V. Lepetit, K. Nishino, and Y. Sato, Eds., vol. 10114, 2017, pp. 136–153, 13th Asian Conference on Computer Vision (ACCV), Taipei, TAIWAN, NOV 20-24, 2016.

[23] Y. Yao, M. Xu, Y. Wang, D. J. Crandall, and E. M. Atkins, "Unsupervised traffic accident detection in first-person videos," in *2019 IEEE/RSJ International Conference on Intelligent Robots and Systems (IROS)*, 2019, pp. 273–280.

[24] W. Bao, Q. Yu, and Y. Kong, "Uncertainty-based traffic accident anticipation with spatio-temporal relational learning," in *Proceedings of the 28th ACM International Conference on Multimedia*, 2020, pp. 2682–2690.

[25] M. Wang, "Smpl annotation tool," https://github.com/wmj142326/SMPL_Tools, 2024.

[26] J. Carreira, P. Agrawal, K. Fragkiadaki, and J. Malik, "Human pose estimation with iterative error feedback," in *2016 IEEE Conference on Computer Vision and Pattern Recognition (CVPR)*, Jun 2016. [Online]. Available: http://dx.doi.org/10.1109/cvpr.2016.512

[27] F. Baradel, R. Brégier, T. Groueix, P. Weinzaepfel, Y. Kalantidis, and G. Rogez, "Posebert: A generic transformer module for temporal 3d human modeling," *IEEE Transactions on Pattern Analysis and Machine Intelligence*, 2022.

[28] C. K. Simms and D. P. Wood, "Effects of pre-impact pedestrian position and motion on kinematics and injuries from vehicle and ground contact," *International Journal of Crashworthiness*, vol. 11, no. 4, pp. 345–355, 2006. [Online]. Available: https://doi.org/10.1533/ijcr.2005.0109

[29] M. Hamacher, L. Eckstein, and R. Paas, "Vehicle related influence of post-car impact pedestrian kinematics on secondary impact," in *Proceedings of the International Research Council on the Biomechanics of Injury conference*, vol. 40. International Research Council on Biomechanics of Injury, 2012, pp. 717–729.

[30] J. Tang, Q. Zhou, B. Nie, T. Yasuki, and Y. Kitagawa, "Influence of pre-impact pedestrian posture on lower extremity kinematics in vehicle collisions," *SAE International journal of transportation safety*, vol. 4, no. 2, pp. 278–288, 2016.

[31] K. Gildea, D. Hall, C. R. Cherry, and C. Simms, "Forward dynamics computational modelling of a cyclist fall with the inclusion of protective response using deep learning-based human pose estimation," *Journal of biomechanics*, vol. 163, p. 111959, 2024.

[32] J. S. Robertson, A. J. Mclean, and G. A. Ryan, "Traffic accidents in adelaide, south australia (1963 - 1964)," *Automobiles*, 1966.

[33] S. Shang, C. Masson, D. Teeling, M. Py, Q. Ferrand, P.-J. Arnoux, and C. Simms, "Kinematics and dynamics of pedestrian head ground contact: A cadaver study," *Safety science*, vol. 127, p. 104684, 2020.

[34] Q. Li, S. Shang, X. Pei, Q. Wang, Q. Zhou, and B. Nie, "Kinetic and kinematic features of pedestrian avoidance behavior in motor vehicle conflicts," *Frontiers in bioengineering and biotechnology*, vol. 9, 2021.

[35] W. Liu, Q. Bao, Y. Sun, and T. Mei, "Recent advances of monocular 2d and 3d human pose estimation: a deep learning perspective," *ACM Computing Surveys*, vol. 55, no. 4, pp. 1–41, 2022.

[36] K. Sun, B. Xiao, D. Liu, and J. Wang, "Deep high-resolution representation learning for human pose estimation," in *2019 IEEE/CVF Conference on Computer Vision and Pattern Recognition (CVPR)*, Jun 2019. [Online]. Available: http://dx.doi.org/10.1109/cvpr.2019.00584

[37] S. Li and A. B. Chan, *3D Human Pose Estimation from Monocular Images with Deep Convolutional Neural Network*, Jan 2015, p. 332–347. [Online]. Available: http://dx.doi.org/10.1007/978-3-319-16808-1_23

[38] G. Pavlakos, X. Zhou, K. G. Derpanis, and K. Daniilidis, "Coarse-to-fine volumetric prediction for single-image 3d human pose," in *2017 IEEE Conference on Computer Vision and Pattern Recognition (CVPR)*, Jul 2017. [Online]. Available: http://dx.doi.org/10.1109/cvpr.2017.139

[39] G. Moon and K. M. Lee, *I2L-MeshNet: Image-to-Lixel Prediction Network for Accurate 3D Human Pose and Mesh Estimation from a Single RGB Image*, Jan 2020, p. 752–768. [Online]. Available: http://dx.doi.org/10.1007/978-3-030-58571-6_44

[40] Z. Qiu, Q. Yang, J. Wang, and D. Fu, "Ivt: An end-to-end instance-guided video transformer for 3d pose estimation," in *Proceedings of the 30th ACM International Conference on Multimedia*, 2022, pp. 6174–6182.

[41] ——, "Dynamic graph reasoning for multi-person 3d pose estimation," in *Proceedings of the 30th ACM International Conference on Multimedia*, 2022, pp. 3521–3529.

[42] C.-H. Chen and D. Ramanan, "3d human pose estimation = 2d pose estimation + matching," in *2017 IEEE Conference on Computer Vision and Pattern Recognition (CVPR)*, Jul 2017. [Online]. Available: http://dx.doi.org/10.1109/cvpr.2017.610

[43] M. Loper, N. Mahmood, J. Romero, G. Pons-Moll, and M. J. Black, "Smpl: a skinned multi-person linear model," *ACM Transactions on Graphics*, p. 1–16, Nov 2015. [Online]. Available: http://dx.doi.org/10.1145/2816795.2818013

[44] F. Bogo, A. Kanazawa, C. Lassner, P. Gehler, J. Romero, and M. J. Black, "Keep it smpl: Automatic estimation of 3d human pose and shape from a single image," in *Computer Vision– ECCV 2016: 14th European Conference, Amsterdam, The Netherlands, October 11-14, 2016, Proceedings, Part V 14*. Springer, 2016, pp. 561–578.

[45] K. Gildea, C. Mercadal-Baudart, R. Blythman, A. Smolic, and C. Simms, "Kinepose: A temporally optimized inverse kinematics technique for 6dof human pose estimation with biomechanical constraints," *arXiv preprint arXiv:2207.12841*, 2022.

[46] W. Luo, W. Liu, and S. Gao, "A revisit of sparse coding based anomaly detection in stacked rnn framework," in *Proceedings of the IEEE international conference on computer vision*, 2017, pp. 341–349.

[47] J. Fang, D. Yan, J. Qiao, J. Xue, and H. Yu, "Dada: Driver attention prediction in driving accident scenarios," *IEEE Transactions on Intelligent Transportation Systems*, vol. 23, no. 6, p. 4959–4971, 2022.

[48] M. Cordts, M. Omran, S. Ramos, T. Rehfeld, M. Enzweiler, R. Benenson, U. Franke, S. Roth, and B. Schiele, "The cityscapes dataset for semantic urban scene understanding," in *Proceedings of the IEEE conference on computer vision and pattern recognition*, 2016, pp. 3213–3223.

[49] S. Zhang, R. Benenson, and B. Schiele, "Citypersons: A diverse dataset for pedestrian detection," in *Proceedings of the IEEE conference on computer vision and pattern recognition*, 2017, pp. 3213–3221.

[50] A. Milan, L. Leal-Taixé, I. Reid, S. Roth, and K. Schindler, "Mot16: A benchmark for multi-object tracking," *arXiv preprint arXiv:1603.00831*, 2016.

[51] M. Andriluka, L. Pishchulin, P. Gehler, and B. Schiele, "2d human pose estimation: New benchmark and state of the art analysis," in *2014 IEEE Conference on Computer Vision and Pattern Recognition*, Jun 2014. [Online]. Available: http://dx.doi.org/10.1109/cvpr.2014.471

[52] U. Iqbal, A. Milan, and J. Gall, "Posetrack: Joint multi-person pose estimation and tracking," in *2017 IEEE Conference on Computer Vision and Pattern Recognition (CVPR)*, Jul 2017. [Online]. Available: http://dx.doi.org/10.1109/cvpr.2017.495

[53] C. Lassner, J. Romero, M. Kiefel, F. Bogo, M. J. Black, and P. V. Gehler, "Unite the people: Closing the loop between 3d and 2d human representations," in *2017 IEEE Conference on Computer Vision and Pattern Recognition (CVPR)*, Jul 2017. [Online]. Available: http://dx.doi.org/10.1109/cvpr.2017.500

[54] H. Joo, N. Neverova, and A. Vedaldi, "Exemplar fine-tuning for 3d human pose fitting towards in-the-wild 3d human pose estimation," *arXiv: Computer Vision and Pattern Recognition,arXiv: Computer Vision and Pattern Recognition*, Apr 2020.

[55] G. Moon and K. Lee, "Neuralannot: Neural annotator for in-the-wild expressive 3d human pose and mesh training sets." Nov 2020.

[56] Y. Tian, H. Zhang, Y. Liu, and L. Wang, "Recovering 3d human mesh from monocular images: A survey," *IEEE transactions on pattern analysis and machine intelligence*, 2023.

[57] J. Fang, J. Qiao, J. Xue, and Z. Li, "Vision-based traffic accident detection and anticipation: A survey," *IEEE Transactions on Circuits and Systems for Video Technology*, 2023.

[58] L. McInnes, J. Healy, and J. Melville, "Umap: Uniform manifold approximation and projection for dimension reduction," *arXiv preprint arXiv:1802.03426*, 2018.

[59] Y. Zhang, P. Sun, Y. Jiang, D. Yu, F. Weng, Z. Yuan, P. Luo, W. Liu, and X. Wang, "Bytetrack: Multi-object tracking by associating every detection box," in *European conference on computer vision*. Springer, 2022, pp. 1–21.

[60] H. Jhuang, J. Gall, S. Zuffi, C. Schmid, and M. J. Black, "Towards understanding action recognition," in *International Conf. on Computer Vision (ICCV)*, Dec. 2013, pp. 3192–3199.

[61] Z. Li, J. Liu, Z. Zhang, S. Xu, and Y. Yan, "Cliff: Carrying location information in full frames into human pose and shape estimation," in *European Conference on Computer Vision*. Springer, 2022, pp. 590–606.

[62] J. Lin, A. Zeng, H. Wang, L. Zhang, and Y. Li, "One-stage 3d whole-body mesh recovery with component aware transformer," in *Proceedings of the IEEE/CVF Conference on Computer Vision and Pattern Recognition*, 2023, pp. 21 159–21 168.

[63] K. He, X. Zhang, S. Ren, and J. Sun, "Deep residual learning for image recognition," in *Proceedings of the IEEE conference on computer vision and pattern recognition*, 2016, pp. 770–778.

[64] Y. Zhou, C. Barnes, J. Lu, J. Yang, and H. Li, "On the continuity of rotation representations in neural networks," in *2019 IEEE/CVF Conference on Computer Vision and Pattern Recognition (CVPR)*, Jun 2019. [Online]. Available: http://dx.doi.org/10.1109/cvpr.2019.00589

[65] A. Paszke, S. Gross, S. Chintala, G. Chanan, E. Yang, Z. DeVito, Z. Lin, A. Desmaison, L. Antiga, and A. Lerer, "Automatic differentiation in pytorch," Oct 2017.

[66] M. Kocabas, N. Athanasiou, and M. J. Black, "Vibe: Video inference for human body pose and shape estimation," in *2020 IEEE/CVF Conference on Computer Vision and Pattern Recognition (CVPR)*, Jun 2020. [Online]. Available: http://dx.doi.org/10.1109/cvpr42600.2020.00530

[67] M. Kocabas, C.-H. P. Huang, O. Hilliges, and M. J. Black, "Pare: Part attention regressor for 3d human body estimation," in *2021 IEEE/CVF International Conference on Computer Vision (ICCV)*, Oct 2021. [Online]. Available: http://dx.doi.org/10.1109/iccv48922.2021.01094

[68] H. Choi, G. Moon, and K. M. Lee, *Pose2Mesh: Graph Convolutional Network for 3D Human Pose and Mesh Recovery from a 2D Human Pose*, Jan 2020, p. 769–787. [Online]. Available: http://dx.doi.org/10.1007/978-3-030-58571-6_45

[69] P. Patel, C.-H. P. Huang, J. Tesch, D. T. Hoffmann, S. Tripathi, and M. J. Black, "Agora: Avatars in geography optimized for regression analysis," in *2021 IEEE/CVF Conference on Computer Vision and Pattern Recognition (CVPR)*, Jun 2021. [Online]. Available: http://dx.doi.org/10.1109/cvpr46437.2021.01326

[70] F. Zhang, X. Zhu, H. Dai, M. Ye, and C. Zhu, "Distribution-aware coordinate representation for human pose estimation," in *2020 IEEE/CVF Conference on Computer Vision and Pattern Recognition (CVPR)*, Jun 2020. [Online]. Available: http://dx.doi.org/10.1109/cvpr42600.2020.00712

[71] J. Brooks, "Coco annotator," https://github.com/jsbroks/coco-annotator/, 2019.

[72] Y. Name, "Human model viewer," https://github.com/Lemon-XQ/human_model_viewer, 2020.

[73] G. Moon, J. Y. Chang, and K. M. Lee, "Camera distance-aware top-down approach for 3d multi-person pose estimation from a single rgb image," in *2019 IEEE/CVF International Conference on Computer Vision (ICCV)*, Oct 2019. [Online]. Available: http://dx.doi.org/10.1109/iccv.2019.01023

[74] C. Zheng, W. Wu, C. Chen, T. Yang, S. Zhu, J. Shen, N. Kehtarnavaz, and M. Shah, "Deep learning-based human pose estimation: A survey," *ACM Computing Surveys*, vol. 56, no. 1, pp. 1–37, 2023.

# Appendix / supplemental material

## Overview

This supplementary material presents more details and additional results not included in the main paper due to page limitation. The list of items included are:

- Details of PVCP Dataset in Sec. A
- Details of the Network Architecture in Sec. B
- Details of Experiment and Results in Sec. C

# A  PVCP Dataset

## A.1  Data Collection.

Our PVCP dataset are all derived from the vehicular perspective of dashcam, and videos are sourced from two primary origins. Similar to previous works (47; 22; 23; 24; 15), we collected videos of pedestrian and vehicle collisions from online platforms such as YouTube, using 'pedestrian-vehicle collision' as a keyword. In addition, a small number of videos are derived from existing open-source traffic datasets (47; 22; 23), which were primarily developed for tasks related to driver attention and the prediction of sudden accidents. From these datasets, we selectively extracted videos that depicted incidents of pedestrian and vehicle collisions. Furthermore, a multi-step filtering process was instituted to ascertain the high quality of the data. Firstly, videos of pedestrian and vehicle collisions within traffic scenarios were selected based on the recorded times and locations. Secondly, we opted for videos shot from a first-person perspective, specifically from the viewpoint of a dashboard camera. Lastly, a resolution analysis was conducted on the videos, those with resolutions too low for human discernment were discarded to ensure the clarity and visibility of the content. All of the collected videos were reduced to individual accident footage, recording a complete pedestrian and vehicle collision, resulting in 209 pedestrian-vehicle accident videos, totaling 42,511 frames of images and about 19,533 pre-collision poses. The statistical results associated with video clips and image frames are shown in Figure A1(a) and Figure A1(b).

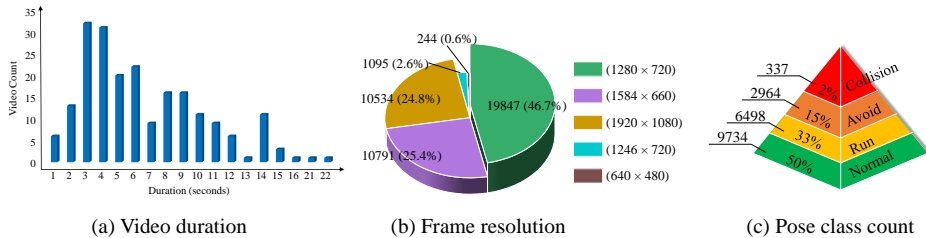

|(a) Video duration|(b) Frame resolution|(c) Pose class count|

Figure A1: PVCP dataset statistics.

## A.2  Data Annotations

In scenarios involving collisions, the pedestrian, the vehicles, and the interconnected data between the two are the pivotal components that must be attend. For the application of pedestrian-vehicle collision accident reconstruction and vehicle active protection in traffic collision scenarios, PVCP provides a rich pedestrian pose representation, including pedestrian clipping images, pedestrian Bounding Box, track id, 2D and 3D keypoints and SMPL mesh label.

### A.2.1  Pose Annotations

**SMPL Annotation Tool**. We used COCO-annotation (71) as a tool for 2D pose annotations and specifically designed an annotation tool (25) for SMPL mesh, which was developed based on (72). By importing the cropped pedestrian background and the initial results estimated by SPIN (11), we ensured that the mesh aligned as closely as possible with the pedestrian's background outline in the image. The final exported mesh model was used as ground truth labels. The annotation interface and

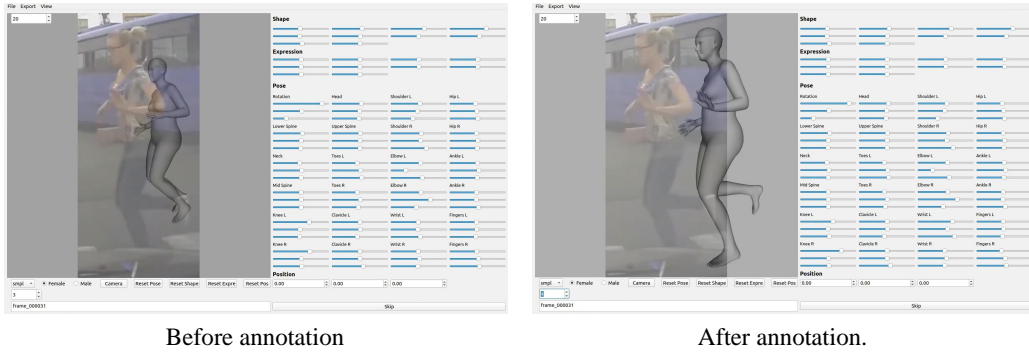

| Before annotation | After annotation. |
|---|---|

Figure A2: Diagram of the SMPL Annotation Tool.

the comparison before and after annotation are shown in the Figure A2. By adjusting the weight of the image background and the mesh foreground, it is clearly displayed to facilitate annotation.

### A.2.2    Vehicle Annotations

Distance between the pedestrian and the vehicle, as well as the vehicle's speed, are crucial factors in collision accidents. In real traffic scenarios, these parameters can often be acquired through the sensors equipped on the vehicle. However, in our collected accident videos, obtaining the distance and speed through sensors is not feasible (as these data are not provided). Therefore, we refer to RootNet (73) to obtain the distance between the pedestrian and the vehicle as our dataset's distance annotation. Similarly, vehicles are often fitted with additional sensors to get accurate speed, but for our crash video dataset it's not perfect and the speed information is missing. Therefore, we use the estimated pedestrian-vehicle distance and video frame rate to obtain an approximate speed label.

### A.2.3    Other Annotations

Beyond the aforementioned annotations, we have also annotated the environment of the collision, the cause of the accident, and whether a collision ultimately occurred. This additional annotation information serves as a supplement to our dataset, facilitating its use in related tasks.

### A.2.4    Visualization Comparison

Compared to existing pose datasets, PVCP demonstrates significant differences and advantages in terms of scene specificity, action space variation, and temporal continuity. As shown in the Figure A3, we visualize a comparison of the differences between PVCP and other pose datasets.

**MSCOCO** Dataset (20) contains annotations for object detection, panoptic segmentation, and keypoint detection. The images are collected from websites including Google, Bing, and Flickr. The annotations are performed by workers on Amazon's Mechanical Turk (AMT). The dataset contains over 200K images and 250K person instances (35). Although the size of COCO datasets for 2D HPE is large enough for normal pose estimation (e.g., standing, walking, running), these datasets have limited training data for unusual poses, such as falling. The data imbalance may cause model bias, resulting in poor performance on those poses (74). Furthermore, the COCO dataset lacks continuous video frames for pose estimation, which presents a challenge for studying sequential pose variations related to temporal features.

**Human3.6M** Dataset (16) is the most widely used multi-view single person 3D human pose benchmark. The dataset is captured in a 4m×3m indoor space using 4 RGB camera, 1 time-of-flight sensor, and 10 motion cameras. It contains 3.6 million 3D human poses and the corresponding videos (50 FPS) in 15 scenarios, such as discussion, sitting on a chair, taking a photo, etc. Especially, both 3D positions and angles of keypoints are available (35). However, these poses belong to everyday activities and differ in spatial and temporal characteristics from specific poses such as pre-collision poses. Moreover, the dataset, captured in an indoor environment, lacks dynamic or static scene backgrounds, which are important factors that can affect the robustness of pose estimation algorithms.

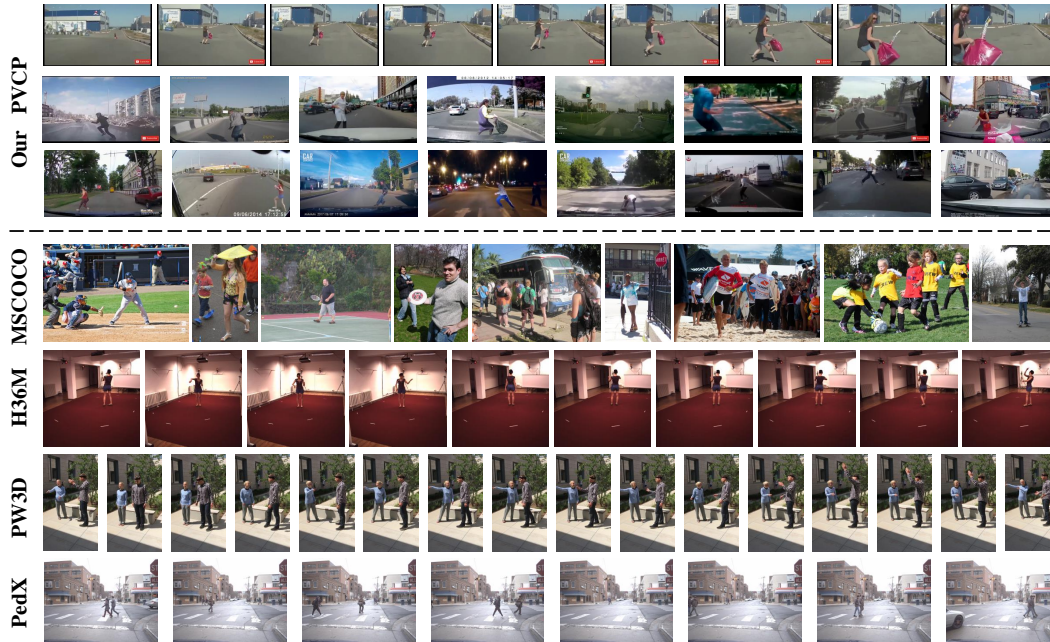

Figure A3: Visualization comparison of PVCP with other pose datasets.

Image background is also one of the key factors influencing the performance of pose estimation methods (56).

**PW3D** Dataset (21) is a single-view multi-person in-the-wild 3D human pose dataset that contains 60 video sequences (24 train, 24 test, and 12 validation) of rich activities, such as climbing, golfing, relaxing on the beach, etc. The videos are captured in various scenes, such as forest, street, playground, shopping mall, etc. They leverage IMU to obtain accurate 3D pose despite the complexity of scenes. Especially, PW3D contains abundant 3D annotations, including 2D/3D pose annotations, 3D body scanning, and SMPL parameters. In some crowded scenes (e.g. on the street), PW3D only provides the label of the target person, ignoring the pedestrians passing by. Generally, the entire dataset is used for evaluation, without any fine-tuning (35). Similar to PW3D, we also annotate pedestrian 2D and 3D keypoints as well as SMPL parameters. However, due to scene constraints, this method of data collection is difficult to replicate in collision scenarios.

**PedX** Dataset (14) is a multi-sensor dataset focused on pedestrian-vehicle interactions at urban intersections. It includes over 5,000 stereo image pairs and 2,500 frames of 3D LiDAR data, all calibrated and time-synchronized. Collected at three four-way intersections, the dataset captures more than 14,000 pedestrian instances from a driver's perspective using roof-mounted stereo cameras. Each pedestrian is annotated with 2D and 3D labels, including 18 keypoints, and unique tracking IDs across frames. Using SMPL parameterization, PedX provides accurate 3D models of pedestrian pose, shape, and global position based on stereo images and LiDAR data. PedX is a pedestrian pose dataset in a traffic scene similar to our PVCP. However, it maintains a fixed viewpoint at intersections, and the pedestrian poses it captures are primarily limited to normal walking postures.

**PVCP** Dataset focuses on dashcam perspective, capturing pedestrians' emergency behaviors during vehicle movement, such as sudden road crossings or running into the street, within dynamic traffic environments and changing fields of view. It highlights rapid, unpredictable motions like accelerating, swerving, or jumping to avoid vehicles. PVCP categorizes these pre-collision poses into four types, and the PPSENet model leverages pose category loss to learn their spatial distinctions. Pedestrian pre-collision poses are closely linked to the vehicle's time series, occurring in rapid succession as the vehicle approaches, emphasizing the importance of time sensitivity and continuity in algorithms. PVCP provides continuous pre-collision emergency poses of pedestrians, this allows for timely pose predictions and warnings by capturing pose variations from a continuous context.

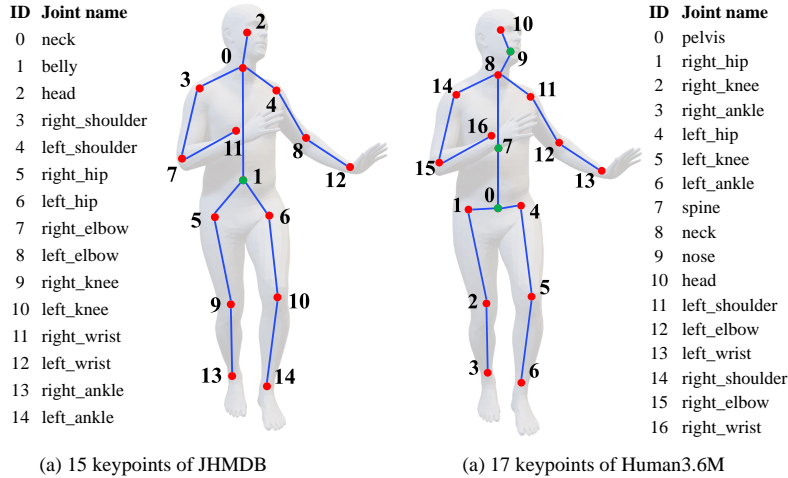

| ID | Joint name |
|----|-----------|
| 0 | neck |
| 1 | belly |
| 2 | head |
| 3 | right_shoulder |
| 4 | left_shoulder |
| 5 | right_hip |
| 6 | left_hip |
| 7 | right_elbow |
| 8 | left_elbow |
| 9 | right_knee |
| 10 | left_knee |
| 11 | right_wrist |
| 12 | left_wrist |
| 13 | right_ankle |
| 14 | left_ankle |

| ID | Joint name |
|----|-----------|
| 0 | pelvis |
| 1 | right_hip |
| 2 | right_knee |
| 3 | right_ankle |
| 4 | left_hip |
| 5 | left_knee |
| 6 | left_ankle |
| 7 | spine |
| 8 | neck |
| 9 | nose |
| 10 | head |
| 11 | left_shoulder |
| 12 | left_elbow |
| 13 | left_wrist |
| 14 | right_shoulder |
| 15 | right_elbow |
| 16 | right_wrist |

(a) 15 keypoints of JHMDB          (a) 17 keypoints of Human3.6M

Figure A4: Pedestrian pose keypoints representation of PVCP dataset.

# B  Network Architecture

## B.1  Image to Pose

Estimating 2D human pose from images is a basic and mature task, and many works have achieved very effective results in different datasets. We take image frame $I$ and corresponding $Bbox$ as input, simply use ResNet50 (63) as the backbone of feature extraction, and use a transposed convolution and a heat map regression head $\mathcal{H}$ as the 2D pose estimation network, which is also the classic paradigm of 2D pose estimation (8). The resulting 2D pose $P_{2D} \in \mathbb{R}^{15 \times 2}$ will be used as input for the next stage. The testset accuracy of ITP is shown in the Table A1. The accuracy of Ankle is significantly lower because the lower limbs of the pedestrian are heavily shielded during the collision.

$$P_{2D} = \mathcal{H}(Deconv(Backbone(I, Bbox))) \tag{A1}$$

Table A1: 2D detected result of ITP.

| Joint | Head | Shoulder | Elbow | Wrist | Hip | Knee | Ankle | Mean |
|-------|------|----------|-------|-------|-----|------|-------|------|
| PCK(%) | 94.82 | 93.62 | 89.61 | 85.69 | 93.05 | 80.05 | 1.93 | 78.15 |

## B.2  Pose to Mesh

We refer to the JHMDB (60) dataset for annotating human 2D pose keypoints (15 keypoints) and participate in the first stage of ITP training. For PTM training, we use the Human3.6M (16) pose representation (17 keypoints). As shown in Figure A4, there are differences in joint positions and numbering between the two. We aligned the common joint positions and calculated the additional joints for Human3.6M:

$$J_{pelvis \mapsto H36M} = (J_{left\_hip \mapsto jhmdb} + J_{right\_hip \mapsto jhmdb}) \times 0.5 \tag{A2}$$

$$J_{spine \mapsto H36M} = (J_{pelvis \mapsto H36M} + J_{neck \mapsto jhmdb}) \times 0.5 \tag{A3}$$

$$J_{nose \mapsto H36M} = (J_{neck \mapsto jhmdb} + J_{head \mapsto jhmdb}) \times 0.5 \tag{A4}$$

Table A2: Component of system. Top use detected 2D pose sequences. Bottom use GT 2D pose sequences.

| Input | Pretrain | Iter | Class Loss | Pose class | MPVE | PAMPVE | MPJPE_14j | PAMPJPE_14j | MPJPE_17j | PAMPJPE_17j |
|---|---|---|---|---|---|---|---|---|---|---|
| | ✓ | | | Normal | 294.73 | 170.10 | 253.80 | 137.39 | 232.74 | 128.24 |
| | | | | Run | 253.16 | 149.99 | 219.06 | 124.01 | 200.19 | 115.27 |
| | | | | Avoid | 286.85 | 159.69 | 246.94 | 124.86 | 222.02 | 114.96 |
| | | | | Collision | 250.58 | 161.25 | 222.47 | 127.37 | 200.38 | 120.47 |
| | | | | All | 282.50 | 163.58 | 243.59 | 132.43 | 222.70 | 123.33 |
| | ✓ | 3 | | Normal | 284.39 | 152.77 | 239.39 | 120.68 | 218.69 | 112.57 |
| | | | | Run | 233.78 | 134.53 | 200.96 | 111.19 | 182.08 | 102.69 |
| | | | | Avoid | 246.02 | 145.64 | 207.84 | 111.54 | 185.50 | 101.94 |
| | | | | Collision | 191.91 | 114.36 | 176.79 | 93.02 | 155.36 | 86.44 |
| 2D Det | | | | All | 266.20 | 146.88 | 225.38 | 116.99 | 204.98 | 108.63 |
| | ✓ | | ✓ | Normal | 275.49 | 149.99 | 233.23 | 119.51 | 213.02 | 111.19 |
| | | | | Run | 224.62 | 129.31 | 193.70 | 108.35 | 174.61 | 100.04 |
| | | | | Avoid | 247.18 | 142.64 | 209.39 | 110.38 | 186.96 | 100.67 |
| | | | | Collision | 291.64 | 124.18 | 263.61 | 104.34 | 231.26 | 99.19 |
| | | | | All | 259.05 | **143.52** | 220.39 | 115.47 | 200.16 | 107.03 |
| | ✓ | 3 | ✓ | Normal | 272.79 | 149.02 | 230.49 | 117.47 | 209.99 | 109.04 |
| | | | | Run | 226.22 | 133.45 | 193.75 | 109.50 | 174.47 | 100.73 |
| | | | | Avoid | 251.60 | 143.52 | 212.75 | 109.75 | 190.00 | 100.09 |
| | | | | Collision | 217.68 | 134.95 | 201.15 | 113.10 | 174.57 | 105.94 |
| | | | | All | **257.75** | 144.19 | **218.61** | **114.50** | **198.16** | **105.86** |
| | ✓ | | | Normal | 156.06 | 103.16 | 132.74 | 80.59 | 120.35 | 74.92 |
| | | | | Run | 129.49 | 89.31 | 109.93 | 70.91 | 100.19 | 65.70 |
| | | | | Avoid | 127.04 | 85.36 | 108.30 | 65.35 | 96.74 | 60.44 |
| | | | | Collision | 135.89 | 89.71 | 127.11 | 70.86 | 112.50 | 64.94 |
| | | | | All | 145.77 | 97.50 | 124.04 | 76.34 | 112.43 | 70.87 |
| | ✓ | 3 | | Normal | 156.21 | 101.34 | 131.47 | 78.38 | 119.50 | 73.13 |
| | | | | Run | 128.03 | 88.85 | 109.57 | 70.81 | 100.04 | 65.66 |
| | | | | Avoid | 129.01 | 89.43 | 107.88 | 67.22 | 96.63 | 61.87 |
| | | | | Collision | 144.93 | 91.80 | 132.99 | 70.39 | 119.26 | 64.53 |
| 2D GT | | | | All | 145.75 | 96.69 | 123.16 | 75.13 | 111.90 | 69.89 |
| | ✓ | | ✓ | Normal | 152.30 | 97.26 | 129.43 | 75.62 | 117.24 | 70.71 |
| | | | | Run | 125.01 | 87.69 | 106.31 | 69.42 | 96.96 | 64.57 |
| | | | | Avoid | 117.76 | 79.36 | 100.16 | 61.37 | 89.80 | 56.92 |
| | | | | Collision | 137.77 | 91.28 | 126.20 | 71.12 | 112.30 | 65.79 |
| | | | | All | 141.28 | **92.78** | 120.16 | **72.43** | 108.90 | **67.58** |
| | ✓ | 3 | ✓ | Normal | 148.58 | 102.71 | 125.34 | 79.05 | 113.42 | 73.21 |
| | | | | Run | 126.73 | 87.65 | 107.87 | 69.62 | 97.97 | 64.52 |
| | | | | Avoid | 126.82 | 86.34 | 106.94 | 66.11 | 95.78 | 60.78 |
| | | | | Collision | 144.13 | 97.48 | 133.35 | 76.46 | 117.56 | 70.12 |
| | | | | All | **140.43** | 96.43 | **118.80** | 75.13 | **107.47** | 69.56 |

## B.3   Loss Function

The loss function of ITP, defined as the Mean Squared Error (MSE), is applied for comparing the predicted heatmaps $\hat{H}$ and the ground truth heatmaps $H$. Following (12), the PTM loss function utilizes SMPL loss and motion loss. Furthermore, The pedestrian pose of the collision sequence has obvious categories, namely normal pose, running pose, avoiding pose and collision pose as described in 3.2. So we introduce pose class loss. The final loss function is as follows:

$$\mathcal{L}_{Class} = \lambda_c \mathcal{L}_{\text{Cross Entropy}}(\hat{C}, C) \tag{A5}$$

$$\mathcal{L}_{PTM} = \mathcal{L}_{SMPL} + \mathcal{L}_{Motion} + \mathcal{L}_{Class} \tag{A6}$$

where $\mathcal{L}_{\text{Cross Entropy}}(\hat{C}, C)$ epresents the cross entropy loss between the predicted pose class and the GT pose class. $\lambda_c$ are the constants of the balance weight loss.

## C   Experimental and Results

### C.1   Evaluation Metric

We evaluated the estimation of 3D human pose and shape using the following metrics, which are briefly described as follows.

**MPJPE**$(mm, \downarrow)$. *Mean Per Joint Position Error* measures the average Euclidean distance between predicted 3D pose and ground truth after root (pelvis joint) matching, which comprehensively evaluates the predicted poses and shapes, including the global rotations.

Table A3: Comparison of 2D GT input in different iterations number.

| Iter | Pose class | MPVE | PAMPVE | MPJPE_14j | PAMPJPE_14j | MPJPE_17j | PAMPJPE_17j |
|---|---|---|---|---|---|---|---|
| | Normal | 150.11 | 103.04 | 126.58 | 79.10 | 114.60 | 73.30 |
| | Run | 128.58 | 88.81 | 109.33 | 70.64 | 99.28 | 65.39 |
| 2 | Avoid | 127.58 | 86.26 | 107.62 | 66.13 | 96.52 | 60.81 |
| | Collision | 143.76 | 95.97 | 134.13 | 75.29 | 117.82 | 69.08 |
| | All | 141.95 | 97.43 | 120.04 | 75.45 | 108.63 | 69.85 |
| | Normal | 148.58 | 102.71 | 125.34 | 79.05 | 113.42 | 73.21 |
| | Run | 126.73 | 87.65 | 107.87 | 69.62 | 97.97 | 64.52 |
| 3 | Avoid | 126.82 | 86.34 | 106.94 | 66.11 | 95.78 | 60.78 |
| | Collision | 144.13 | 97.48 | 133.35 | 76.46 | 117.56 | 70.12 |
| | All | 140.43 | **96.43** | 118.80 | **75.13** | 107.47 | **69.56** |
| | Normal | 148.11 | 102.79 | 124.99 | 79.21 | 113.11 | 73.38 |
| | Run | 126.12 | 87.22 | 107.46 | 69.25 | 97.61 | 64.23 |
| 4 | Avoid | 126.66 | 86.74 | 106.80 | 66.40 | 95.58 | 61.06 |
| | Collision | 144.69 | 98.67 | 133.11 | 77.44 | 117.68 | 71.05 |
| | All | **139.96** | 96.92 | **118.46** | 75.19 | **107.16** | 69.62 |
| | Normal | 148.17 | 103.04 | 125.08 | 79.51 | 113.24 | 73.66 |
| | Run | 126.08 | 87.11 | 107.50 | 69.14 | 97.67 | 64.17 |
| 5 | Avoid | 126.83 | 87.28 | 106.94 | 66.82 | 95.69 | 61.48 |
| | Collision | 145.48 | 99.73 | 133.25 | 78.33 | 118.12 | 71.92 |
| | All | 140.01 | 97.10 | 118.54 | 75.40 | 107.27 | 69.83 |
| | Normal | 148.59 | 103.39 | 125.45 | 79.88 | 113.67 | 74.02 |
| | Run | 126.40 | 87.19 | 107.82 | 69.18 | 98.03 | 64.26 |
| 6 | Avoid | 127.27 | 87.93 | 107.29 | 67.34 | 96.04 | 62.00 |
| | Collision | 146.56 | 100.73 | 133.70 | 79.21 | 118.88 | 72.80 |
| | All | 140.41 | 97.42 | 118.89 | 75.70 | 107.68 | 70.14 |

Table A4: Comparison of results on PVCP testset after training other datasets.

| Input | Train Set | Test Set | Pose class | MPVE | PAMPVE | MPJPE_14j | PAMPJPE_14j | MPJPE_17j | PAMPJPE_17j |
|---|---|---|---|---|---|---|---|---|---|
| | | | Normal | 191.45 | 118.21 | 167.38 | 92.63 | 153.22 | 87.62 |
| | | | Run | 182.26 | 119.20 | 157.14 | 93.66 | 143.76 | 86.64 |
| | COCO | PVCP | Avoid | 160.62 | 99.18 | 136.47 | 78.21 | 124.86 | 72.87 |
| | | | Collision | 165.36 | 90.39 | 150.89 | 71.81 | 138.62 | 65.10 |
| | | | All | 185.57 | 116.28 | 161.28 | 91.24 | 147.61 | 85.65 |
| | | | Normal | 206.31 | 134.79 | 175.23 | 106.68 | 159.88 | 98.70 |
| | | | Run | 181.16 | 129.40 | 154.96 | 104.39 | 139.40 | 94.35 |
| 2D GT | PW3D | PVCP | Avoid | 178.75 | 110.68 | 150.77 | 85.23 | 137.80 | 78.03 |
| | | | Collision | 154.24 | 90.50 | 141.18 | 77.98 | 127.11 | 71.19 |
| | | | All | 196.33 | 130.50 | 167.01 | 103.61 | 151.86 | 95.17 |
| | | | Normal | 148.58 | 102.71 | 125.34 | 79.05 | 113.42 | 73.21 |
| | | | Run | 126.73 | 87.65 | 107.87 | 69.62 | 97.97 | 64.52 |
| | PVCP | PVCP | Avoid | 126.82 | 86.34 | 106.94 | 66.11 | 95.78 | 60.78 |
| | (finetuning) | | Collision | 144.13 | 97.48 | 133.35 | 76.46 | 117.56 | 70.12 |
| | | | All | **140.43** | **96.43** | **118.80** | **75.13** | **107.47** | **69.56** |

**PA-MPJPE**($mm, \downarrow$). *Procrustes-Aligned Mean Per Joint Position Error* denotes MPJPE after rigid alignment of the predicted 3D pose and ground truth, which eliminates the discrepancies in scale and global rotation.

**MPVE**($mm, \downarrow$). *Mean Per-vertex Error*, initially aligns similarly to MPJPE. It is characterized by the average point-to-point Euclidean distance between predicted mesh vertices and corresponding ground truth vertices.

**PA-MPVE**($mm, \downarrow$). *Procrustes-Aligned Mean Per-vertex Error* represents MPVE after applying a Procrustes alignment between predicted mesh vertices and ground truth vertices, similar to how PA-MPJPE is calculated.

Further, we test the errors of 14 keypoints ($X\_14j$) shared by 2D and 3D pose representations (21) and 17 keypoints ($X\_17j$) represented only by 3D pose representations (16) respectively. For 2d pose estimation, we use PCK as metrics.

**PCK**($\%, \uparrow$). *Percentage of Correct Keypoints* measures the accuracy of joint positioning in the body. If the candidate body joint falls within the threshold pixel of the GT joint, it is considered correct.

## C.2 Ablation Study

**Component of system.** In the stage of lifting 2D poses to 3D meshes, we incorporated an iterative decoder. Instead of predicting the output in a single step, we iteratively refined the results to gradually approach the optimal solution. Additionally, we included a pose classification detection head and used a class loss as supervision. We compared the impact of different components on the network's performance, reloading and training the model each time, and set the optimal number of iterations to 3. Under the combined effect of these two modules, the pose sequences from both types of input achieved minimal error. As shown in Table A2, we provide a detailed comparison of the errors for different pre-collision pose class. The iterative decoder significantly improved the accuracy across all pose categories, and the introduction of the pose category loss function further enhanced the accuracy of pre-collision poses.

**Number of iterations.** As shown in the Table A3, we combined all modules to verify the accuracy of different iterations of 2D GT input to select the optimal number of iterations. We verify the test results of increasing the number of iterations from 2 to 6 in turn, and it can be seen that the test results of different iterations are excellent. When *iter*=3 or 4, the average error of each metric is the lowest value. We chose *iter*=3 as the final number of iterations for experimental comparison.

**Without Fine-Tuning.** As shown in the Table A4, we trained on the COCO and PW3D dataset respectively and validated on the PVCP testset. Because these datasets do not have pose category annotations, we set them uniformly to a single category. The results show that training only on another large human pose dataset without fine-tuning for PVCP does not perform well on PVCP this particular pose dataset. The conclusion that can be drawn is that training solely on other large human pose datasets, such as COCO and PW3D, without fine-tuning for PVCP, does not yield good performance on the PVCP dataset. This indicates that there are significant differences between datasets, particularly when dealing with specific scenarios and unique poses, and that fine-tuning and optimization tailored to the PVCP dataset are necessary to improve model performance.

